# How Receptive Field Parameters Affect Neural Learning

**Bartlett W. Mel**
CNS Program
Caltech, 216-76
Pasadena, CA 91125

**Stephen M. Omohundro**
ICSI
1947 Center St., Suite 600
Berkeley, CA 94704

## Abstract

We identify the three principle factors affecting the performance of learning by networks with localized units: unit noise, sample density, and the structure of the target function. We then analyze the effect of unit receptive field parameters on these factors and use this analysis to propose a new learning algorithm which dynamically alters receptive field properties during learning.

## 1  LEARNING WITH LOCALIZED RECEPTIVE FIELDS

Locally-tuned representations are common in both biological and artificial neural networks. Several workers have analyzed the effect of receptive field size, shape, and overlap on representation accuracy: (Baldi, 1988), (Ballard, 1987), and (Hinton, 1986). This paper investigates the additional interactions introduced by the task of function learning. Previous studies which have considered learning have for the most part restricted attention to the use of the input probability distribution to determine receptive field layout (Kohonen, 1984) and (Moody and Darken, 1989). We will see that the structure of the function being learned may also be advantageously taken into account.

Function learning using radial basis functions (RBF's) is currently a popular technique (Broomhead and Lowe, 1988) and serves as an adequate framework for our discussion. Because we are interested in constraints on biological systems, we must explictly consider the effects of unit noise. The goal is to choose the layout of receptive fields so as to minimize average performance error.

Let $y = f(\mathbf{x})$ be the function the network is attempting to learn from example

$(\mathbf{x}, y)$ pairs. The network consists of $N$ units whose locally-tuned receptive fields are distributed across the input space. The activity of the $i$th unit is the sum of a radial basis function $\phi_i(\mathbf{x})$ and a mean-zero noise process $\eta_i(\mathbf{x})$. A typical form for $\phi_i$ is an $n$-dimensional Gaussian parametrized by its center $\mathbf{x}_i$ and width $\sigma_i$,

$$\phi_i(\mathbf{x}) = e^{\frac{-\|\mathbf{x}_i - \mathbf{x}\|^2}{2\sigma_i^2}}. \tag{1}$$

The function $f(\mathbf{x})$ is approximated as a weighted sum of the output of $N$ of these units:

$$F(\mathbf{x}) = \sum_{i=1}^{N} w_i[\phi_i(\mathbf{x}) + \eta_i(\mathbf{x})]. \tag{2}$$

The weights $w_i$ are trained using the LMS (least mean square) rule, which attempts to minimize the mean squared distance between $f$ and $F$ over the set of training patterns $p$ for the current layout of receptive fields. In the next section we address the additional considerations that arise when the receptive field centers and sizes are allowed to vary in addition to the weights.

## 2    TWO KINDS OF ERROR

To understand the effect of receptive field properties on performance we must distinguish two basic sources of error. The first we call *estimation error* and is due to the intrinsic unit noise. The other we call *approximation error* and arises from the inability of the unit activity functions to represent the target function.

### 2.1    ESTIMATION ERROR

The *estimation error* can be characterized by the variance in $F(\mathbf{x}) \mid \mathbf{x}$. Because of the intrinsic unit noise, repeated stimulation of a network with the same input vector $\mathbf{x}_0$ will generate a distribution of outputs $F(\mathbf{x}_0)$. If this variance is large, it can be a significant contribution to the MSE (fig. 1). Consideration of noisy units is most relevant to biological networks and analog hardware implementations of artificial units. Averaging is a powerful statistical technique for reducing the variance of a distribution. In the current context, averaging corresponds to receptive field overlap. In general, the more overlap the better the noise reduction in $F(\mathbf{x})$ (though see section 2.2). The overlap of units at $\mathbf{x}_0$ can be increased by either increasing the density of receptive field centers there, or broadening the receptive fields of units in the neighborhood.

From equation 2, $F(\mathbf{x})$ may be rewritten

$$F(\mathbf{x}) = \sum_{i=1}^{N} \phi_i(\mathbf{x})w_i + \xi(\mathbf{x}), \tag{3}$$

where the summation term is the noise-free LMS approximation to $f(\mathbf{x})$, and the second term

$$\xi(\mathbf{x}) = \sum_{i=1}^{N} \eta_i(\mathbf{x})w_i, \tag{4}$$

## Joint Density

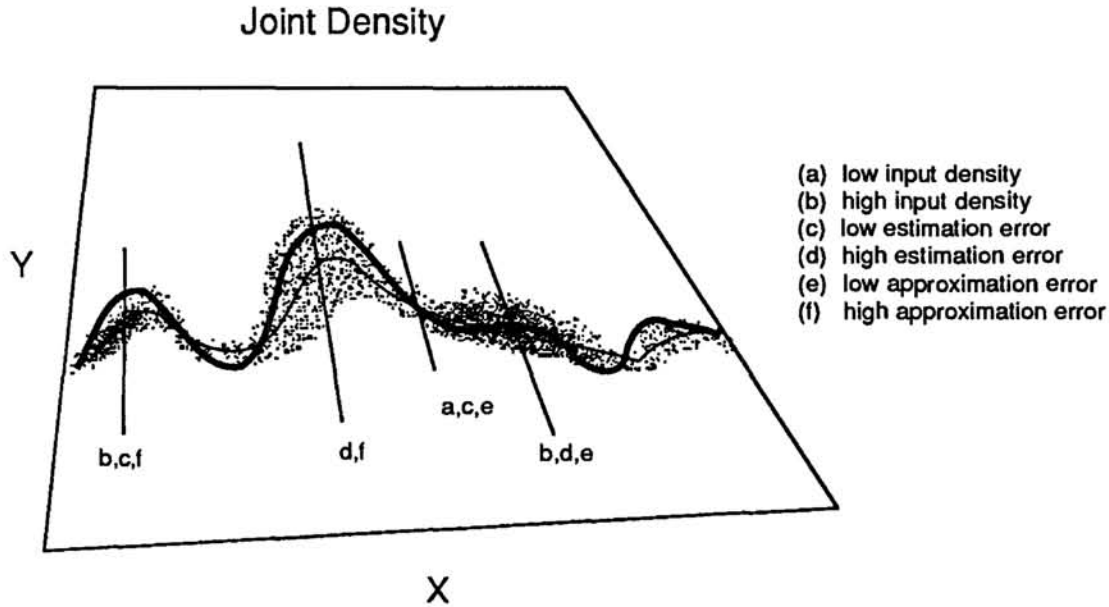

(a)  low input density
(b)  high input density
(c)  low estimation error
(d)  high estimation error
(e)  low approximation error
(f)  high approximation error

Figure 1: A. Estimation error arises from the variance of $F(\mathbf{x}) \mid \mathbf{x}$. B. Approximation error is the deviation of the mean from the desired response $(f(\mathbf{x}) - <F(\mathbf{x})>)^2$.

is the estimation error. Since $\xi(\mathbf{x})$ has mean zero for all $\mathbf{x}$, its variance is

$$\text{Var}[\xi(\mathbf{x})] = \text{E}[\xi^2(\mathbf{x})] = \text{E}[\sum_{i=1}^{N}\eta_i^2(\mathbf{x})w_i^2]. \tag{5}$$

If each unit has the same noise profile, this reduces to

$$\text{Var}[\xi] = \text{Var}[\eta] \sum_{i=1}^{N} w_i^2. \tag{6}$$

The dependence of estimation error $\xi$ on the size of weights explains why increasing the density of receptive fields in the input space reduces noise in the output of the learning network. Though the *number* of units, and hence weights, that contribute to the output is increased in this manipulation, the estimation error is proportional to the sum of *squared* weights (6). The benefit achieved by making weights smaller outruns the cost of increasing their number. For example, each receptive field with weight $w_i$ may be replaced by two copies of itself with weight $w_i/2$ and leave $F(\mathbf{x})$ unchanged. The new sum of squared weights, $\sum_{i=1}^{N} 2(\frac{w_i}{2})^2$, and hence the estimation error, is reduced by a factor of two, however.

A second strategy that may lead to a reduction in the size of weights involves *broadening* receptive fields (see section 2.2 for conditions). In general, broadening receptive fields increases the unweighted output of the network $\sum_{i=1}^{N} \phi_i(\mathbf{x})$, implying that the weights $w_i$ must be correspondingly reduced in order that $\| F(\mathbf{x}) \|$ remain approximately constant.

These observations suggest that the effects of noise are best mitigated by allocating receptive field resources in regions of the input space where units are heavily weighted. It is interesting to note that under the assumption of additive noise, the functional form $\phi$ of the receptive fields themselves has no direct effect on the estimation error in $F(\mathbf{x})$. The response profiles may, however, *indirectly* affect estimation error via the weight vector, since LMS weights on receptive fields of different functional forms will generally be different.

## 2.2    APPROXIMATION ERROR

The second fundamental type of error, which we call *approximation error*, persists even for noise-free input units, and is due to error in the "fit" of the approximating function $F$ to the target function $f$ (fig. 1). Two aspects of approximation error are distinguished in the following sections.

### 2.2.1    MISMATCH OF FUNCTIONAL FORM

First, there may be mismatch between the specific functional form of the basis functions and that of the target function. For example, errors naturally arise when linear RBF's are used to approximate nonlinear target functions, since curves cannot be perfectly fit with straight lines. However, these errors may be made vanishingly small by increasing the density of receptive fields. For example, if linear receptive fields are trained to best fit a curved region of $f(\mathbf{x})$ with second derivative $c$, then the mean squared error, $\int_{-d/2}^{d/2} (\frac{c}{2}x^2 - a)^2$ has a value $O(c^2 d^5)$. This type of error falls off as the 5th power of $d$, where $d$ is the spacing of the receptive fields. In a similar result, (Baldi and Heilegenberg, 1988) show that approximations to both linear and quadratic functions improve exponentially fast with increasing density of Gaussian receptive fields.

### 2.2.2    MISMATCH OF SPATIAL SCALE

A more general source of error in fitting target functions occurs when receptive fields are either too broad or too widely spaced relative to the fine spatial structure of $f$. Both of these factors can act to locally limit the high frequency content of the approximation $F$, which may give rise to severe approximation errors.

The Nyquist (and Shannon) result on signal sampling says that the highest frequency which may be recovered from a sampled signal is half the sampling frequency. If the receptive field density is not high enough then this kind of result shows that high frequency fine structure in the function being approximated will be lost.

When the unit receptive fields are excessively wide, they can also wash out the high frequency fine structure of the function. One can think of $F$ as a "blurred" version of the the weight vector which in turn is a sampled version of $f$. The blurring is greater for wide receptive fields. The density and width should be chosen to match their frequency transfer characteristics and best approximate the function. For one-dimensional Gaussian receptive fields of width $\sigma$, we choose the receptive

field spacing $d$ to be

$$d = \frac{\pi}{2}\sigma. \tag{7}$$

A density that satisfies this type of condition will be referred to in the next section as a "frequency-matched" density.

# 3  A RECEPTIVE FIELD DESIGN STRATEGY

In this section we describe an adaptive learning strategy based on the results above. Figure 2 shows the results of an experimental implementation of this procedure.

It is possible to empirically measure the magnitude of the two sources of error analyzed above. Since we wish to minimize the expected performance error for the network as a whole, we weight our measurements of each type of error at each $\mathbf{x}$ by the input probability $\rho(\mathbf{x})$. Errors in high density regions count more. Small magnitude errors may be important in high probability regions while even large errors may be neglected in low probability regions. The learning algorithm adjusts the layout of receptive fields to adjust to each form of error in turn. The steps involved follow.

1. Uniformly distribute broad receptive fields at frequency-matched density throughout regions of the input space that contain data. (In our 1-d example, data, and hence receptive fields, are present across the entire domain.)

2. Train the network weights to an LMS solution with fixed receptive fields. Using the trained network, accrue approximation errors across the input space.

3. Where the approximation error exceeds a threshold $\tau$ anywhere within a unit's receptive field, split the receptive field into two subfields that are as small as possible while still locally maintaining frequency-matched density. (This depends on receptive field profile). Repeat steps 2 and 3 until the approximation error is under threshold across entire input space. We now have a layout where receptive field width and density are locally matched to the spatial frequency content of the target function, and approximation error is small and uniform across the input space. Note that since errors accrue according to $\rho(\mathbf{x})$, we have preferentially allocated resources (through splitting) in regions with both high error and high input probability.

4. Using the current network, measure and accrue estimation errors across the input space.

5. Where the estimation error exceeds $\tau$ anywhere within a unit's receptive field, replace the receptive field by two of the same size, adding a small random pertubation to each center. Repeat from 4 until estimation error is below threshold across entire input space. We now have a layout where receptive field *density* is highest where the effects of noise were most severe, such that estimation error is now small and uniform across the input space. Once again, we have preferentially allocated resources in regions with both high error and high input probability.

Figure 2 illustrates this process for a noisy, one-dimensionsal learning problem. Each frame shows the estimation error, the approximation error, the target func-

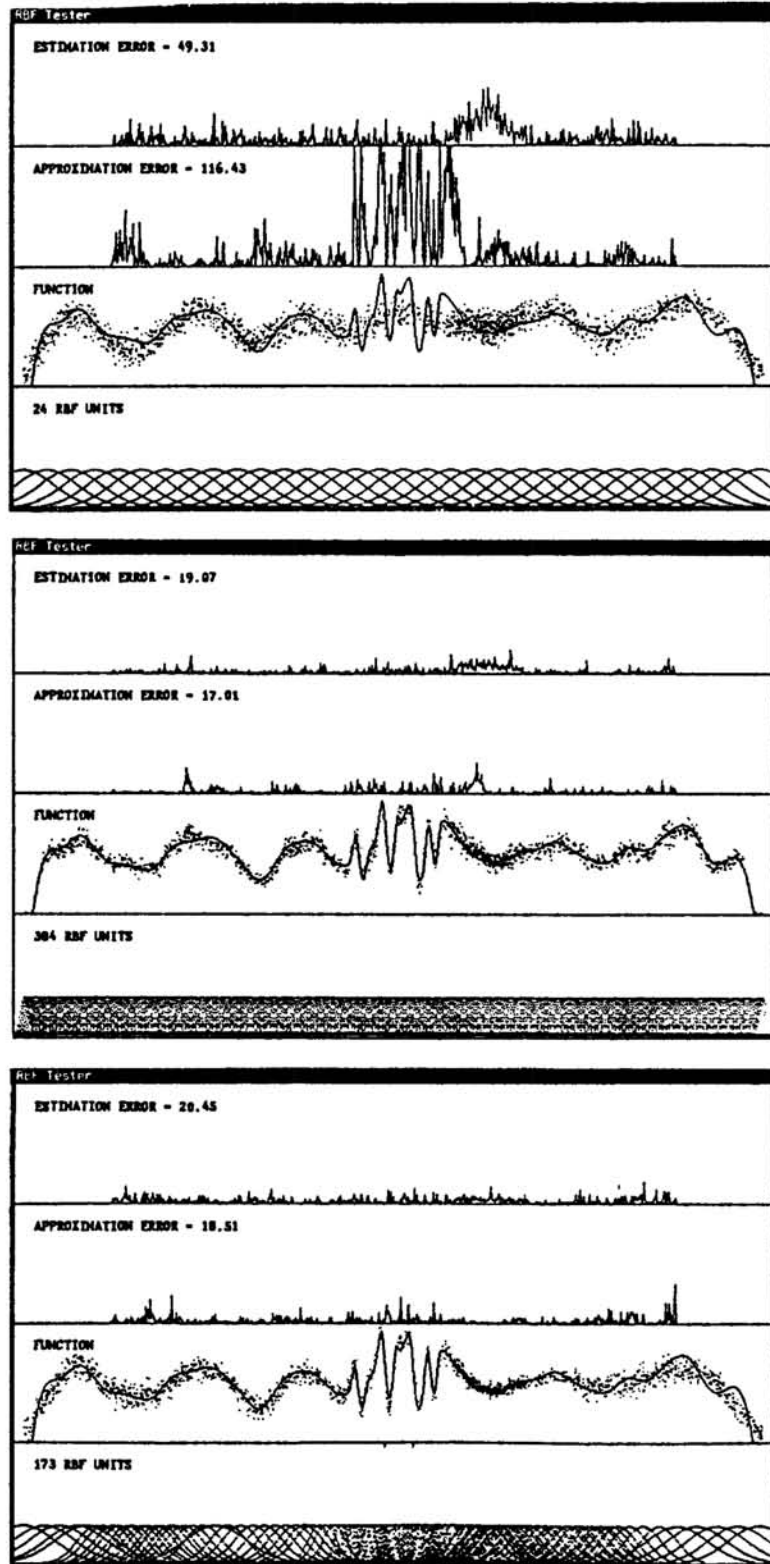

Figure 2: Results of an adaptive strategy for choosing receptive field size and density. See text for details.

tion and network output, and the unit response functions. In the top frame 24 units with broad, noisy receptive fields have been LMS-trained to fit the target function. Estimation error is visible across the entire domain, though it is concentrated in the small region just to the right of center where the input probability is peaked. Approximation error is concentrated in the central region which contains high spatial frequencies, with minor secondary peaks in other regions, including the region of high input probability.

In the second frame, the receptive field width was uniformly decreased and density was uniformly increased to the point where MSE fell below $\tau$; 384 units were required. In the third frame, the adaptive strategy presented above was used to allocate units and choose widths. Fewer than half as many units (173) were needed in this example to achieve the same MSE as in the second frame. In higher dimensions, and with sparser data, this kind of recursive splitting and doubling strategy should be even more important.

## 4   CONCLUSIONS

In this paper we have shown how receptive field size, shape, density, and noise characteristics interact with the frequency content of target functions and input probability density to contribute to both estimation and approximation errors during supervised function learning. Based on these interrelationships, a simple, adaptive, error-driven strategy for laying out receptive fields was demonstrated that makes efficient use of unit resources in the attempt to minimize mean squared performance error.

An improved understanding of the role of receptive field structure in learning may in the future help in the interpretation of patterns of coarse-coding seen in many biological sensory and motor systems.

### References

Baldi, P. & Heiligengerg, W. How sensory maps could enhance resolution through ordered arrangements of broadly tuned receptors. *Biol. Cybern.*, 1988, *59*, 313-318.

Ballard, D.H. Interpolation coding: a representation for numbers in neural models. *Biol. Cybern.*, 1987, *57*, 389-402.

Broomhead, D.S. & Lowe, D. Multivariable functional interpolation and adaptive networks. *Complex Systems*, 1988, *2*, 321-355.

Hinton, G.E. (1986) Distributed representations. In *Parallel distributed processing: explorations in the microstructure of cognition, vol. 1*, D.E. Rumelhart, J.L. McClelland, (Eds.), Bradford, Cambridge.

Kohonen, T. *Self organization and associative memory*. Springer-Verlag: Berlin, 1984.

MacKay, D. Hyperacuity and coarse-coding. In preparation.

Moody, J. & Darken, C. Fast learning in networks of locally-tuned processing units. *Neural Computation*, 1989, *1*, 281-294.


